# Predictive Approaches For Choosing Hyperparameters in Gaussian Processes

**S. Sundararajan**
Computer Science and Automation
Indian Institute of Science
Bangalore 560 012, India
*sundar@csa.iisc.ernet.in*

**S. Sathiya Keerthi**
Mechanical and Production Engg.
National University of Singapore
10 Kentridge Crescent, Singapore 119260
*mpessk@guppy.mpe.nus.edu.sg*

## Abstract

Gaussian Processes are powerful regression models specified by parametrized mean and covariance functions. Standard approaches to estimate these parameters (known by the name Hyperparameters) are Maximum Likelihood (ML) and Maximum APosterior (MAP) approaches. In this paper, we propose and investigate predictive approaches, namely, maximization of Geisser's Surrogate Predictive Probability (GPP) and minimization of mean square error with respect to GPP (referred to as Geisser's Predictive mean square Error (GPE)) to estimate the hyperparameters. We also derive results for the standard Cross-Validation (CV) error and make a comparison. These approaches are tested on a number of problems and experimental results show that these approaches are strongly competitive to existing approaches.

## 1  Introduction

Gaussian Processes (GPs) are powerful regression models that have gained popularity recently, though they have appeared in different forms in the literature for years. They can be used for classification also; see MacKay (1997), Rasmussen (1996) and Williams and Rasmussen (1996). Here, we restrict ourselves to regression problems. Neal (1996) showed that a large class of neural network models converge to a Gaussian Process prior over functions in the limit of an infinite number of hidden units. Although GPs can be created using infinite networks, often GPs are specified directly using parametric forms for the mean and covariance functions (Williams and Rasmussen (1996)). We assume that the process is zero mean. Let $\mathbf{Z}_N = \{\mathbf{X}_N, \mathbf{y}_N\}$ where $\mathbf{X}_N = \{\mathbf{x}(i): i = 1, \ldots, N\}$ and $\mathbf{y}_N = \{\mathbf{y}(i): i = 1, \ldots, N\}$. Here, $\mathbf{y}(i)$ represents the output corresponding to the input vector $\mathbf{x}(i)$. Then, the Gaussian prior over the functions is given by

$$p(\mathbf{y}_N|\mathbf{X}_N, \bar{\boldsymbol{\theta}}) = \frac{exp(-\mathbf{y}_N^T \bar{\mathbf{C}}_N^{-1} \mathbf{y}_N)}{(2\pi)^{\frac{N}{2}} |\bar{\mathbf{C}}_N|^{\frac{1}{2}}} \tag{1}$$

where $\bar{\mathbf{C}}_N$ is the covariance matrix with $(i,j)^{th}$ element $[\bar{\mathbf{C}}_N]_{i,j} = \bar{C}(\mathbf{x}(i), \mathbf{x}(j); \bar{\boldsymbol{\theta}})$ and $\bar{C}(.; \bar{\boldsymbol{\theta}})$ denotes the parametrized covariance function. Now, assuming that the

observed output $\mathbf{t}_N$ is modeled as $\mathbf{t}_N = \mathbf{y}_N + \boldsymbol{\epsilon}_N$ and $\boldsymbol{\epsilon}_N$ is zero mean multivariate Gaussian with covariance matrix $\sigma^2 \mathbf{I}_N$ and is independent of $\mathbf{y}_N$, we get

$$p(\mathbf{t}_N | \mathbf{X}_N, \boldsymbol{\theta}) = \frac{exp(-\mathbf{t}_N^T \mathbf{C}_N^{-1} \mathbf{t}_N)}{(2\pi)^{\frac{N}{2}} |\mathbf{C}_N|^{\frac{1}{2}}} \qquad (2)$$

where $\mathbf{C}_N = \bar{\mathbf{C}}_N + \sigma^2 \mathbf{I}_N$. Therefore, $[\mathbf{C}_N]_{i,j} = [\bar{\mathbf{C}}_N]_{i,j} + \sigma^2 \delta_{i,j}$, where $\delta_{i,j} = 1$ when $i = j$ and zero otherwise. Note that $\boldsymbol{\theta} = (\bar{\boldsymbol{\theta}}, \sigma^2)$ is the new set of hyperparameters. Then, the predictive distribution of the output $y(N+1)$ for a test case $\mathbf{x}(N+1)$ is also Gaussian with mean and variance

$$\hat{y}(N+1) = \mathbf{k}_{N+1}^T \mathbf{C}_N^{-1} \mathbf{t}_N \qquad (3)$$

and

$$\sigma_{y(N+1)}^2 = b_{N+1} - \mathbf{k}_{N+1}^T \mathbf{C}_N^{-1} \mathbf{k}_{N+1} \qquad (4)$$

where $b_{N+1} = C(\mathbf{x}(N+1), \mathbf{x}(N+1); \boldsymbol{\theta})$ and $\mathbf{k}_{N+1}$ is an $N \times 1$ vector with $i^{th}$ element given by $C(\mathbf{x}(N+1), \mathbf{x}(i); \boldsymbol{\theta})$. Now, we need to specify the covariance function $C(.; \boldsymbol{\theta})$. Williams and Rasmussen (1996) found the following covariance function to work well in practice.

$$\bar{C}(\mathbf{x}(i), \mathbf{x}(j); \bar{\boldsymbol{\theta}}) = a_0 + a_1 \sum_{p=1}^{M} x_p(i) x_p(j) + v_0 exp(-\frac{1}{2} \sum_{p=1}^{M} w_p (x_p(i) - x_p(j))^2) \quad (5)$$

where $x_p(i)$ is the $p^{th}$ component of $i^{th}$ input vector $\mathbf{x}(i)$. The $w_p$ are the Automatic Relevance Determination (ARD) parameters. Note that $C(\mathbf{x}(i), \mathbf{x}(j); \boldsymbol{\theta}) = \bar{C}(\mathbf{x}(i), \mathbf{x}(j); \bar{\boldsymbol{\theta}}) + \sigma^2 \delta_{i,j}$. Also, all the parameters are positive and it is convenient to use logarithmic scale. Hence, $\boldsymbol{\theta}$ is given by $log(a_0, a_1, v_0, w_1, \ldots, w_M, \sigma^2)$. Then, the question is: how do we handle $\boldsymbol{\theta}$ ? More sophisticated techniques like Hybrid Monte Carlo (HMC) methods (Rasmussen (1996) and Neal (1997)) are available which can numerically integrate over the hyperparameters to make predictions. Alternately, we can estimate $\boldsymbol{\theta}$ from the training data. We restrict to the latter approach here. In the classical approach, $\boldsymbol{\theta}$ is assumed to be deterministic but unknown and the estimate is found by maximizing the likelihood (2). That is, $\boldsymbol{\theta}_{ML} = \overset{argmax}{\boldsymbol{\theta}} \ p(\mathbf{t}_N | \mathbf{X}_N, \boldsymbol{\theta})$. In the Bayesian approach, $\boldsymbol{\theta}$ is assumed to be random and a prior $p(\boldsymbol{\theta})$ is specified. Then, the MAP estimate $\boldsymbol{\theta}_{MP}$ is obtained as $\boldsymbol{\theta}_{MP} = \overset{argmax}{\boldsymbol{\theta}} \ p(\mathbf{t}_N | \mathbf{X}_N, \boldsymbol{\theta}) p(\boldsymbol{\theta})$ with the motivation that the the predictive distribution $p(y(N+1) | \mathbf{x}(N+1), \mathbf{Z}_N)$ can be approximated as $p(y(N+1) | \mathbf{x}(N+1), \mathbf{Z}_N, \boldsymbol{\theta}_{MP})$. With this background, in this paper we propose and investigate different predictive approaches to estimate the hyperparameters from the training data.

## 2 Predictive approaches for choosing hyperparameters

Geisser (1975) proposed Predictive Sample Reuse (PSR) methodology that can be applied for both model selection and parameter estimation problems. The basic idea is to define a partition scheme $P(N, n, \Gamma)$ such that $P_{N-n}^{(i)} = (\mathbf{Z}_{N-n}^{ir}; \mathbf{Z}_n^{io})$ is $i^{th}$ partition belonging to a set $\Gamma$ of partitions with $\mathbf{Z}_{N-n}^{ir}, \mathbf{Z}_n^{io}$ representing the $N - n$ retained and $n$ omitted data sets respectively. Then, the unknown $\boldsymbol{\theta}$ is estimated (or a model $M_j$ is chosen among a set of models indexed by $j = 1, \ldots, J$) by means of optimizing a predictive measure that measures the predictive performance on the omitted observations $\mathbf{X}_n^{io}$ by using the retained observations $\mathbf{Z}_{N-n}^{ir}$ averaged over the partitions ($i \in \Gamma$). In the special case of $n = 1$, we have the leave one out strategy. Note that this approach was independently presented in the

name of cross-validation (CV) by Stone (1974). The well known examples are the standard CV error and negative of average predictive likelihood. Geisser and Eddy (1979) proposed to maximize $\prod_{i=1}^{N} p(t(i)|\mathbf{x}(i), \mathbf{Z}_N^{(i)}, M_j)$ (known as Geisser's surrogate Predictive Probability (GPP)) by synthesizing Bayesian and PSR methodology in the context of (parametrized) model selection. Here, we propose to maximize $\prod_{i=1}^{N} p(t(i)|\mathbf{x}(i), \mathbf{Z}_N^{(i)}, \boldsymbol{\theta})$ to estimate $\boldsymbol{\theta}$, where $\mathbf{Z}_N^{(i)}$ is obtained from $\mathbf{Z}_N$ by removing the $i^{th}$ sample. Note that $p(t(i)|\mathbf{x}(i), \mathbf{Z}_N^{(i)}, \boldsymbol{\theta})$ is nothing but the predictive distribution $p(y(i)|\mathbf{x}(i), \mathbf{Z}_N^{(i)}, \boldsymbol{\theta})$ evaluated at $y(i) = t(i)$. Also, we introduce the notion of Geisser's Predictive mean square Error (GPE) defined as $\frac{1}{N} \sum_{i=1}^{N} E((y(i) - t(i))^2)$ (where the expectation operation is defined with respect to $p(y(i)|\mathbf{x}(i), \mathbf{Z}_N^{(i)}, \boldsymbol{\theta})$) and propose to estimate $\boldsymbol{\theta}$ by minimizing GPE.

## 2.1 Expressions for GPP and its gradient

The objective function corresponding to GPP is given by

$$G(\boldsymbol{\theta}) = -\frac{1}{N} \sum_{i=1}^{N} log(p(t(i)|\mathbf{x}(i), \mathbf{Z}_N^{(i)}, \boldsymbol{\theta}) \tag{6}$$

From (3) and (4) we get

$$G(\boldsymbol{\theta}) = \frac{1}{N} \sum_{i=1}^{N} \frac{(t(i) - \hat{y}(i))^2}{2\sigma_{y(i)}^2} + \frac{1}{2N} \sum_{i=1}^{N} log\, \sigma_{y(i)}^2 + \frac{1}{2} log\, 2\pi \tag{7}$$

where $\hat{y}(i) = [\mathbf{c}_i^{(i)}]^T [\mathbf{C}_N^{(i)}]^{-1} \mathbf{t}_N^{(i)}$ and $\sigma_{y(i)}^2 = c_{ii} - [\mathbf{c}_i^{(i)}]^T [\mathbf{C}_N^{(i)}]^{-1} \mathbf{c}_i^{(i)}$. Here, $\mathbf{C}_N^{(i)}$ is an $N-1 \times N-1$ matrix obtained from $\mathbf{C}_N$ by removing the $i^{th}$ column and $i^{th}$ row. Similarly, $\mathbf{t}_N^{(i)}$ and $\mathbf{c}_i^{(i)}$ are obtained from $\mathbf{t}_N$ and $\mathbf{c}_i$ (i.e., $i^{th}$ column of $\mathbf{C}_N$) respectively by removing the $i^{th}$ element. Then, $G(\boldsymbol{\theta})$ and its gradient can be computed efficiently using the following result.

**Theorem 1** *The objective function $G(\boldsymbol{\theta})$ under the Gaussian Process model is given by*

$$G(\boldsymbol{\theta}) = \frac{1}{2N} \sum_{i=1}^{N} \frac{q_N^2(i)}{\bar{c}_{ii}} - \frac{1}{2N} \sum_{i=1}^{N} log\bar{c}_{ii} + \frac{1}{2} log 2\pi \tag{8}$$

*where $\bar{c}_{ii}$ denotes the $i^{th}$ diagonal entry of $\mathbf{C}_N^{-1}$ and $q_N(i)$ denotes the $i^{th}$ element of $\mathbf{q}_N = \mathbf{C}_N^{-1} \mathbf{t}_N$. Its gradient is given by*

$$\frac{\partial G(\boldsymbol{\theta})}{\partial \theta_j} = \frac{1}{2N} \sum_{i=1}^{N} \left(1 + \frac{q_N^2(i)}{\bar{c}_{ii}}\right)\left(\frac{s_{j,i}}{\bar{c}_{ii}}\right) + \frac{1}{N} \sum_{i=1}^{N} q_N(i)\left(\frac{r_j(i)}{\bar{c}_{ii}}\right) \tag{9}$$

*where $s_{j,i} = \bar{\mathbf{c}}_i^T \frac{\partial \mathbf{C}_N}{\partial \theta_j} \bar{\mathbf{c}}_i$, $\mathbf{r}_j = -\mathbf{C}_N^{-1} \frac{\partial \mathbf{C}_N}{\partial \theta_j} \mathbf{C}_N^{-1} \mathbf{t}_N$ and $\mathbf{q}_N = \mathbf{C}_N^{-1} \mathbf{t}_N$. Here, $\bar{\mathbf{c}}_i$ denotes the $i^{th}$ column of the matrix $\mathbf{C}_N^{-1}$.*

Thus, using (8) and (9) we can compute the GPP and its gradient. We will give meaningful interpretation to the different terms shortly.

## 2.2 Expressions for CV function and its gradient

We define the CV function as

$$H(\boldsymbol{\theta}) = \frac{1}{N} \sum_{i=1}^{N} (t(i) - \hat{y}(i))^2 \tag{10}$$

where $\hat{y}(i)$ is the mean of the conditional predictive distribution as given above. Now, using the following result we can compute $H(\boldsymbol{\theta})$ efficiently.

**Theorem 2** *The CV function $H(\boldsymbol{\theta})$ under the Gaussian model is given by*

$$H(\boldsymbol{\theta}) = \frac{1}{N} \sum_{i=1}^{N} \left( \frac{q_N(i)}{\bar{c}_{ii}} \right)^2 \tag{11}$$

*and its gradient is given by*

$$\frac{\partial H(\boldsymbol{\theta})}{\partial \theta_j} = \frac{1}{N} \sum_{i=1}^{N} \left( \frac{2}{\bar{c}_{ii}} \right) \left( \frac{q_N(i) r_j(i)}{\bar{c}_{ii}} + \left( \frac{q_N^2(i)}{\bar{c}_{ii}} \right) \left( \frac{s_{j,i}}{\bar{c}_{ii}} \right) \right) \tag{12}$$

*where $s_{j,i}, \mathbf{r}_j, q_N(i)$ and $\bar{c}_{ii}$ are as defined in theorem 1.*

### 2.3 Expressions for GPE and its gradient

The GPE function is defined as

$$G_E(\boldsymbol{\theta}) = \frac{1}{N} \sum_{i=1}^{N} \int (t(i) - y(i))^2 \, p(y(i)|\mathbf{x}(i), \mathbf{Z}_N^{(i)}, \boldsymbol{\theta}) \, dy(i) \tag{13}$$

which can be readily simplified to

$$G_E(\boldsymbol{\theta}) = \frac{1}{N} \sum_{i=1}^{N} (t(i) - \hat{y}(i))^2 + \frac{1}{N} \sum_{i=1}^{N} \sigma_{y(i)}^2 \tag{14}$$

On comparing (14) with (10), we see that while CV error minimizes the deviation from the predictive mean, GPE takes predictive variance also into account. Now, the gradient can be written as

$$\frac{\partial G_E(\boldsymbol{\theta})}{\partial \theta_j} = \frac{\partial H(\boldsymbol{\theta})}{\partial \theta_j} + \frac{1}{N} \sum_{i=1}^{N} \left( \frac{1}{\bar{c}_{ii}} \right)^2 \bar{\mathbf{c}}_i^T \frac{\partial \mathbf{C}_N}{\partial \theta_j} \bar{\mathbf{c}}_i \tag{15}$$

where we have used the results $\sigma_{y(i)}^2 = \frac{1}{\bar{c}_{ii}}$, $\frac{\partial \bar{c}_{ii}}{\partial \theta_j} = \mathbf{e}_i^T \frac{\partial \mathbf{C}_N^{-1}}{\partial \theta_j} \mathbf{e}_i$ and $\frac{\partial \mathbf{C}_N^{-1}}{\partial \theta_j} = -\mathbf{C}_N^{-1} \frac{\partial \mathbf{C}_N}{\partial \theta_j} \mathbf{C}_N^{-1}$. Here $\mathbf{e}_i$ denotes the $i^{th}$ column vector of the identity matrix $\mathbf{I}_N$.

### 2.4 Interpretations

More insight can be obtained from reparametrizing the covariance function as follows.

$$C(\mathbf{x}(i), \mathbf{x}(j); \boldsymbol{\theta}) = \sigma^2 \left( \bar{a}_0 + \bar{a}_1 \sum_{p=1}^{M} \mathbf{x}_p(i) \mathbf{x}_p(j) + \bar{v}_0 exp(-\frac{1}{2} \sum_{p=1}^{M} w_p (\mathbf{x}_p(i) - \mathbf{x}_p(j))^2) + \delta_{i,j} \right) \tag{16}$$

where $a_0 = \sigma^2 \bar{a}_0$, $a_1 = \sigma^2 \bar{a}_1$, $v_0 = \sigma^2 \bar{v}_0$. Let us define $P(\mathbf{x}(i), \mathbf{x}(j); \boldsymbol{\theta}) = \frac{1}{\sigma^2} C(\mathbf{x}(i), \mathbf{x}(j); \boldsymbol{\theta})$. Then $\mathbf{P}_N^{-1} = \sigma^2 \mathbf{C}_N^{-1}$. Therefore, $\bar{c}_{i,j} = \frac{\bar{p}_{i,j}}{\sigma^2}$ where $\bar{c}_{i,j}, \bar{p}_{i,j}$ denote the $(i, j)^{th}$ element of the matrices $\mathbf{C}_N^{-1}$ and $\mathbf{P}_N^{-1}$ respectively. From theorem 2 (see (10) and (11)) we have $t(i) - \hat{y}(i) = \frac{q_N(i)}{\bar{c}_{ii}} = \frac{\bar{\mathbf{c}}_i^T \mathbf{t}_N}{\bar{c}_{ii}}$. Then, we can rewrite (8) as

$$G(\boldsymbol{\theta}) = \frac{1}{2N\sigma^2} \sum_{i=1}^{N} \frac{\bar{q}_N^2(i)}{\bar{p}_{ii}} - \frac{1}{2N} \sum_{i=1}^{N} log \bar{p}_{ii} + \frac{1}{2} log 2\pi\sigma^2 \tag{17}$$

Here, $\bar{q}_N = \mathbf{P}_N^{-1} \mathbf{t}_N$ and, $\bar{p}_i$, $\bar{p}_{ii}$ denote, respectively, the $i^{th}$ column and $i^{th}$ diagonal entry of the matrix $\mathbf{P}_N^{-1}$. Now, by setting the derivative of (17) with respect to $\sigma^2$ to zero, we can infer the noise level as

$$\hat{\sigma}^2 = \frac{1}{N} \sum_{i=1}^{N} \frac{\bar{q}_N^2(i)}{\bar{p}_{ii}} \qquad (18)$$

Similarly, the CV error (10) can be rewritten as

$$H(\boldsymbol{\theta}) = \frac{1}{N} \sum_{i=1}^{N} \frac{\bar{q}_N^2(i)}{\bar{p}_{ii}^2} \qquad (19)$$

Note that $H(\boldsymbol{\theta})$ is dependent only on the ratio of the hyperparameters (i.e., on $\bar{a}_0, \bar{a}_1, \bar{v}_0$) apart from the ARD parameters. Therefore, we cannot infer the noise level uniquely. However, we can estimate the ARD parameters and the ratios $\bar{a}_0, \bar{a}_1, \bar{v}_0$. Once we have estimated these parameters, then we can use (18) to estimate the noise level. Next, we note that the noise level preferred by the GPE criterion is zero. To see this, first let us rewrite (14) under reparametrization as

$$G_E(\boldsymbol{\theta}) = \frac{1}{N} \sum_{i=1}^{N} \frac{\bar{q}_N^2(i)}{\bar{p}_{ii}^2} + \frac{\sigma^2}{N} \sum_{i=1}^{N} \frac{1}{\bar{p}_{ii}} \qquad (20)$$

Since $\bar{q}_N(i)$ and $\bar{p}_{ii}$ are independent of $\sigma^2$, it follows that the GPE prefers zero as the noise level, which is not true. Therefore, this approach can be applied when, either the noise level is known or a good estimate of it is available.

## 3    Simulation results

We carried out simulation on four data sets. We considered MacKay's robot arm problem and its modified version introduced by Neal (1996). We used the same data set as MacKay (2-inputs and 2-outputs), with 200 examples in the training set and 200 in the test set. This data set is referred to as 'data set 1' in Table 1. Next, to evaluate the ability of the predictive approaches in estimating the ARD parameters, we carried out simulation on the robot arm data with 6 inputs (Neal's version), denoted as 'data set 2' in Table 1. This data set was generated by adding four further inputs, two of which were copies of the two inputs corrupted by additive zero mean Gaussian noise of standard deviation 0.02 and two further irrelevant Gaussian noise inputs with zero mean and unit variance (Williams and Rasmussen (1996)). The performance measures chosen were average of Test Set Error (normalized by true noise level of 0.0025) and average of negative logarithm of predictive probability (NLPP) (computed from Gaussian density function with (3) and (4)). Friedman's [1] data sets 1 and 2 were based on the problem of predicting impedance and phase respectively from four parameters of an electrical circuit. Training sets of three different sizes (50, 100, 200) and with a signal-to-noise ratio of about 3:1 were replicated 100 times and for each training set (at each sample size N), scaled integral squared error ($ISE = \frac{\int_D (y(\mathbf{x}) - \hat{y}(\mathbf{x}))^2 d\mathbf{x}}{var_D\, y(\mathbf{x})}$) and NLPP were computed using 5000 data points randomly generated from a uniform distribution over $D$ (Friedman (1991)). In the case of GPE (denoted as $G_E$ in the tables), we used the noise level estimate generated from Gaussian distribution with mean $NL_T$ (true noise level) and standard deviation $0.03\, NL_T$. In the case of CV, we estimated the hyperparameters in the reparametrized form and estimated the noise level using (18). In the case of MAP (denoted as $MP$ in the tables), we used the same prior

Table 1: Results on robot arm data sets. Average of normalized test set error (TSE) and negative logarithm of predictive probability (NLPP) for various methods.

|       | Data Set : 1 | | Data Set : 2 | |
|-------|-------|--------|-------|--------|
|       | TSE   | NLPP   | TSE   | NLPP   |
| $ML$  | 1.126 | -1.512 | 1.131 | -1.512 |
| $MP$  | 1.131 | -1.511 | 1.181 | -1.489 |
| $G_P$ | 1.115 | -1.524 | 1.116 | -1.516 |
| $CV$  | 1.112 | -1.518 | 1.146 | -1.514 |
| $G_E$ | 1.111 | -1.524 | 1.112 | -1.524 |

Table 2: Results on Friedman's data sets. Average of scaled integral squared error and negative logarithm of predictive probability (given in brackets) for different training sample sizes and various methods.

|       | Data Set : 1 | | | Data Set : 2 | | |
|-------|------------|------------|------------|------------|------------|------------|
|       | N = 50     | N = 100    | N = 200    | N = 50     | N = 100    | N = 200    |
| $ML$  | 0.43(7.24) | 0.19(6.71) | 0.10(6.49) | 0.26(1.05) | 0.16(0.82) | 0.11(0.68) |
| $MP$  | 0.42(7.18) | 0.22(6.78) | 0.12(6.56) | 0.25(1.01) | 0.16(0.82) | 0.11(0.69) |
| $G_P$ | 0.47(7.29) | 0.20(6.65) | 0.10(6.44) | 0.33(1.25) | 0.20(0.86) | 0.12(0.70) |
| $CV$  | 0.55(7.27) | 0.22(6.67) | 0.10(6.44) | 0.42(1.36) | 0.21(0.91) | 0.13(0.70) |
| $G_E$ | 0.35(7.10) | 0.15(6.60) | 0.08(6.37) | 0.28(1.20) | 0.18(0.85) | 0.12(0.63) |

given in Rasmussen (1996). The GPP approach is denoted as $G_P$ in the tables. For all these methods, conjugate gradient (CG) algorithm (Rasmussen (1996)) was used to optimize the hyperparameters. The termination criterion (relative function error) with a tolerance of $10^{-7}$ was used, but with a constraint on the maximum number of CG iterations set to 100. In the case of robot arm data sets, the algorithm was run with ten different initial conditions and the best solution (chosen from respective best objective function value) is reported. The optimization was carried out separately for the two outputs and the results reported are the average TSE, NLPP. In the case of Friedman's data sets, the optimization algorithm was run with three different initial conditions and the best solution was picked up. When $N = 200$, the optimization algorithm was run with only one initial condition. For all the data sets, both the inputs and outputs were normalized to zero mean and unit variance.

From Table 1, we see that the performances (both TSE and NLPP) of the predictive approaches are better than ML and MAP approaches for both the data sets. In the case of data set 2, we observed that like ML and MAP methods, all the predictive approaches rightly identified the irrelevant inputs. The performance of GPE approach is the best on the robot arm data and demonstrates the usefulness of this approach when a good noise level estimate is available. In the case of Friedman's data set 1 (see Table 2), the important observation is that the performances (both ISE and NLPP) of GPP, CV approaches are relatively poor at low sample size ($N = 50$) and improve very well as $N$ increases. Note that the performances of the predictive approaches are better compared to the ML and MAP methods starting from $N = 100$ onwards (see NLPP). Again, GPE gives the best performance and the performance at low sample size ($N = 50$) is also quite good. In the case of Friedman's data set 2, the ML and MAP approaches perform better compared to the predictive approaches except $GPE$. The performances of GPP and CV improve

as $N$ increases and are very close to the ML and MAP methods when $N = 200$. Next, it is clear that the MAP method gives the best performance at low sample size. This behavior, we believe, is because the prior plays an important role and hence is very useful. Also, note that unlike data set 1, the performance of $GPE$ is inferior to ML and MAP approaches at low sample sizes and improves over these approaches (see NLPP) as $N$ increases. This suggests that the knowledge of the noise level alone is not the only issue. The basic issue we think is that the predictive approaches estimate the predictive performance of a given model from the training samples. Clearly, the quality of the estimate will become better as $N$ increases. Also, knowing the noise level improves the quality of the estimate.

## 4 Discussion

Simulation results indicate that the size $N$ required to get good estimates of predictive performance will be dependent on the problem. When $N$ is sufficiently large, we find that the predictive approaches perform better than ML and MAP approaches. The *sufficient* number of samples can be as low as 100 as evident from our results on Friedman's data set 1. Also, MAP approach is the best, when $N$ is very low. As one would expect, the performances of ML and MAP approaches are nearly same as $N$ increases. The comparison with the existing approaches indicate that the predictive approaches developed here are strongly competitive. The overall cost for computing the function and the gradient (for all three predictive approaches) is $O(MN^3)$. The cost for making prediction is same as the one required for ML and MAP methods. The proofs of the results and detailed simulation results will be presented in another paper (Sundararajan and Keerthi, 1999).

## References

Friedman, J.H., (1991) Multivariate Adaptive Regression Splines, *Ann. of Stat.*, **19**, 1-141.

Geisser, S., (1975) The Predictive Sample Reuse Method with Applications, *Journal of the American Statistical Association*, **70**, 320-328.

Geisser, S., and Eddy, W.F., (1979) A Predictive Approach to Model Selection, *Journal of the American Statistical Association*, **74**, 153-160.

MacKay, D.J.C. (1997) Gaussian Processes - *A replacement for neural networks ?*, Available in Postscript via *URL http://www.wol.ra.phy.cam.ac.uk/mackay/*.

Neal, R.M. (1996) *Bayesian Learning for Neural Networks*, New York: Springer-Verlag.

Neal, R.M. (1997) Monte Carlo Implementation of Gaussian Process Models for Bayesian Regression and Classification. Tech. Rep. No. 9702, Dept. of Statistics, University of Toronto.

Rasmussen, C. (1996) *Evaluation of Gaussian Processes and other Methods for Non-Linear Regression*, Ph.D. Thesis, Dept. of Computer Science, University of Toronto.

Stone, M. (1974) Cross-Validatory Choice and Assessment of Statistical Predictions (with discussion), *Journal of Royal Statistical Society, ser.B*, **36**, 111-147.

Sundararajan, S., and Keerthi, S.S. (1999) Predictive Approaches for Choosing Hyperparameters in Gaussian Processes, submitted to *Neural Computation*, available at: *http://guppy.mpe.nus.edu.sg/~mpessk/gp/gp.html*.

Williams, C.K.I., and Rasmussen, C.E. (1996) Gaussian Processes for Regression. In *Advances in Neural Information Processing Systems 8*, ed. by D.S.Touretzky, M.C.Mozer, and M.E.Hasselmo. MIT Press.